# Inverting Grice's Maxims to Learn Rules from Natural Language Extractions

**Mohammad Shahed Sorower, Thomas G. Dietterich, Janardhan Rao Doppa**
**Walker Orr, Prasad Tadepalli, and Xiaoli Fern**
School of Electrical Engineering and Computer Science
Oregon State University
Corvallis, OR 97331
{sorower,tgd,doppa,orr,tadepall,xfern}@eecs.oregonstate.edu

## Abstract

We consider the problem of learning rules from natural language text sources. These sources, such as news articles and web texts, are created by a writer to communicate information to a reader, where the writer and reader share substantial domain knowledge. Consequently, the texts tend to be concise and mention the minimum information necessary for the reader to draw the correct conclusions. We study the problem of learning domain knowledge from such concise texts, which is an instance of the general problem of learning in the presence of missing data. However, unlike standard approaches to missing data, in this setting we know that facts are more likely to be missing from the text in cases where the reader can infer them from the facts that are mentioned combined with the domain knowledge. Hence, we can explicitly model this "missingness" process and invert it via probabilistic inference to learn the underlying domain knowledge. This paper introduces a *mention model* that models the probability of facts being mentioned in the text based on what other facts have already been mentioned and domain knowledge in the form of Horn clause rules. Learning must simultaneously search the space of rules and learn the parameters of the mention model. We accomplish this via an application of Expectation Maximization within a Markov Logic framework. An experimental evaluation on synthetic and natural text data shows that the method can learn accurate rules and apply them to new texts to make correct inferences. Experiments also show that the method out-performs the standard EM approach that assumes mentions are missing at random.

## 1 Introduction

The immense volume of textual information available on the web provides an important opportunity and challenge for AI: Can we develop methods that can learn domain knowledge by reading natural texts such as news articles and web pages. We would like to acquire at least two kinds of domain knowledge: concrete facts and general rules. Concrete facts can be extracted as logical relations or as tuples to populate a data base. Systems such as Whirl [3], TextRunner [5], and NELL [1] learn extraction patterns that can be applied to text to extract instances of relations.

General rules can be acquired in two ways. First, they may be stated explicitly in the text—particularly in tutorial texts. Second, they can be acquired by generalizing from the extracted concrete facts. In this paper, we focus on the latter setting: Given a data base of literals extracted from natural language texts (e.g., newspaper articles), we seek to learn a set of probabilistic Horn clauses that capture general rules.

Unfortunately for rule learning algorithms, natural language texts are incomplete. The writer tends to mention only enough information to allow the reader to easily infer the remaining facts from shared background knowledge. This aspect of economy in language was first pointed out by Grice

[7] in his maxims of cooperative conversation (see Table 1). For example, consider the following sentence that discusses a National Football League (NFL) game:

*"Given the commanding lead of Kansas city on the road, Denver Broncos' 14-10 victory surprised many"*

This mentions that Kansas City is the away team and that the Denver Broncos won the game, but does not mention that Kansas City lost the game or that the Denver Broncos was the home team. Of course these facts can be inferred from domain knowledge rules such as the rule that "if one team is the winner, the other is the loser (and vice versa)" and the rule "if one team is the home team, the other is the away team (and vice versa)". This is an instance of the second maxim.

Table 1: Grice's Conversational Maxims

| | |
|---|---|
| 1 | Be truthful—do not say falsehoods. |
| 2 | Be concise—say as much as necessary, but no more. |
| 3 | Be relevant. |
| 4 | Be clear. |

Another interesting case arises when shared knowledge could lead the reader to an incorrect inference:

*"Ahmed Said Khadr, an Egyptian-born Canadian, was killed last October in Pakistan."*

This explicitly mentions that Khadr is Canadian, because otherwise the reader would infer that he was Egyptian based on the domain knowledge rule "if a person is born in a country, then the person is a citizen of that country". Grice did not discuss this case, but we can state this as a corollary of the first maxim: Do not by omission mislead the reader into believing falsehoods.

This paper formalizes the first two maxims, including this corollary, and then shows how to apply them to learn probabilistic Horn clause rules from propositions extracted from news stories. We show that rules learned this way are able to correctly infer more information from incomplete texts than a baseline approach that treats propositions in news stories as missing at random.

The problem of learning rules from extracted texts has been studied previously [11, 2, 17]. These systems rely on finding documents in which all of the facts participating in a rule are mentioned. If enough such documents can be found, then standard rule learning algorithms can be applied. A drawback of this approach is that it is difficult to learn rules unless there are many documents that provide such complete training examples. The central hypothesis of our work is that by explicitly modeling the process by which facts are mentioned, we can learn rules from sets of documents that are smaller and less complete.

The line of work most similar to this paper is that of Michael and Valiant [10, 9] and Doppa, et al. [4]. They study learning hard (non-probabilistic) rules from incomplete extractions. In contrast with our approach of learning explicit probabilistic models, they take the simpler approach of implicitly inverting the conversational maxims when counting evidence for a proposed rule. Specifically, they count an example as consistent with a proposed rule unless it explicitly contradicts the rule. Although this approach is much less expensive than the probabilistic approach described in this paper, it has difficulty with soft (probabilistic) rules. To handle these, these authors sort the rules by their scores and keep high scoring rules even if they have some contradictions. Such an approach can learn "almost hard" rules, but will have difficulty with rules that are highly probabilistic (e.g., that the home team is somewhat more likely to win a game than the away team).

Our method has additional advantages. First, it provides a more general framework that can support alternative sets of conversational maxims, such as mentions based on saliency, recency (prefer to mention a more recent event rather than an older event), and surprise (prefer to mention a less likely event rather than a more likely event). Second, when applied to new articles, it assigns probabilities to alternative interpretations, which is important for subsequent processing. Third, it provides an elegant, first-principles account of the process, which can then be compiled to yield more efficient learning and reasoning procedures.

## 2   Technical Approach

We begin with a logical formalization of the Gricean maxims. Then we present our implementation of these maxims in Markov Logic [15]. Finally, we describe a method for probabilistically inverting the maxims to learn rules from textual mentions.

**Formalizing the Gricean maxims.** Consider a writer and a reader who share domain knowledge $K$. Suppose that when told a fact $F$, the reader will infer an additional fact $G$. We will write this as $(K, \text{MENTION}(F) \vdash_{reader} G)$, where $\vdash_{reader}$ represents the inference procedure of the reader and MENTION is a modal operator that captures the action of mentioning a fact in the text. Note that the reader's inference procedure is not standard first-order deduction, but instead is likely to be incomplete and non-monotonic or probabilistic.

With this notation, we can formalize the first two Gricean maxims as follows:

- Mention true facts/don't lie:

$$F \quad \Rightarrow \quad \text{MENTION}(F) \tag{1}$$
$$\text{MENTION}(F) \quad \Rightarrow \quad F \tag{2}$$

  The first formula is overly strong, because it requires the writer to mention all true facts. Below, we will show how to use Markov Logic weights to weaken this. The second formula captures a positive version of "don't lie"—if something is mentioned, then it is true. For news articles, it does not need to be weakened probabilistically.

- Don't mention facts that can be inferred by the reader:

$$\text{MENTION}(F) \wedge G \wedge (K, \text{MENTION}(F) \vdash_{reader} G \Rightarrow \neg\text{MENTION}(G)$$

- Mention facts needed to prevent incorrect reader inferences:

$$\text{MENTION}(F) \wedge \neg G \wedge (K, \text{MENTION}(F) \vdash_{reader} G) \wedge$$
$$H \wedge (K, \text{MENTION}(F \wedge H) \nvdash_{reader} G) \quad \Rightarrow \quad \text{MENTION}(H)$$

  In this formula $H$ is a true fact that, when combined with $F$, is sufficient to prevent the reader from inferring $G$.

**Implementation in Markov Logic.** Although this formalization is very general, it is difficult to apply directly because of the embedded invocation of the reader's inference procedure and the use of the MENTION modality. Consequently, we sidestep this problem by manually "compiling" the maxims into ordinary first-order Markov Logic as follows. The notation $w$ : indicates that a rule has a weight $w$ in Markov Logic.

The first maxim is encoded in terms of fact-to-mention and mention-to-fact rules. For each predicate $P$ in the domain of discourse, we write

$$w_1 : \text{FACT\_P} \quad \Rightarrow \quad \text{MENTION\_P}$$
$$w_2 : \text{MENTION\_P} \quad \Rightarrow \quad \text{FACT\_P}.$$

Suppose that the shared knowledge $K$ contains the Horn clause rule $P \Rightarrow Q$, then we encode the positive form of second maxim in terms of the mention-to-mention rule:

$$w_3 : \text{MENTION\_P} \wedge \text{FACT\_Q} \quad \Rightarrow \quad \neg\text{MENTION\_Q}$$

One might expect that we could encode the faulty-inference-by-omission corollary as

$$w_4 : \text{MENTION\_P} \wedge \neg\text{FACT\_Q} \quad \Rightarrow \quad \text{MENTION\_NOTQ},$$

where we have chosen MENTION_NOTQ to play the role of $H$ in axiom 2. However, in news stories, there is a strong preference for $H$ to be a positive assertion, rather than a negative assertion. For example, in the citizenship case, it would be unnatural to say *"Ahmed Said Khadr, an Egyptian-born non-Egyptian..."*. In particular, because $\text{CITIZENOF}(p, c)$ is generally a function from $p$ to $c$ (i.e., a person is typically a citizen of only one country), it suffices to mention $\text{CITIZENOF}(Khadr, Canada)$ to prevent the faulty inference $\text{CITIZENOF}(Khadr, Egypt)$. Hence, for rules of the form $P(x, y) \Rightarrow Q(x, y)$, where $Q$ is a function from its first to its second argument, we can implement the inference-by-omission maxim as

$$w_5 : \text{MENTION\_P}(x, y) \wedge \text{FACT\_Q}(x, z) \wedge (y \neq z) \quad \Rightarrow \quad \text{MENTION\_Q}(x, z).$$

Finally, the shared knowledge $P \Rightarrow Q$ is represented by the fact-to-fact rule:

$$w_6 : \text{FACT\_P} \quad \Rightarrow \quad \text{FACT\_Q}$$

In Markov Logic, each of these rules is assigned a (learned) weight which can be viewed as a cost of violating the rule. The probability of a world $\omega$ is proportional to

$$\exp\left(\sum_j w_j I[\text{Rule } j \text{ is satisfied by } \omega]\right),$$

where $j$ iterates over all groundings of the Markov logic rules in world $\omega$ and $I[\phi]$ is 1 if $\phi$ is true and 0 otherwise.

An advantage of Markov Logic is that it allows us to define a probabilistic model even when there are contradictions and cycles in the logical rules. Hence, we can include both a rule that says "if the home team is mentioned, then the away team is not mentioned" and rules that say "the home team is always mentioned" and "the away team is always mentioned". Obviously a possible world $\omega$ cannot satisfy all of these rules. The relative weights on the rules determine the probability that particular literals are actually mentioned.

**Learning.** We seek to learn both the rules and their weights. We proceed by first proposing candidate fact-to-fact rules and then automatically generating the other rules (especially the mention-to-mention rules) from the general rule schemata described above. Then we apply EM to learn the weights on all of the rules. This has the effect of removing unnecessary rules by driving their weights to zero.

**Proposing Candidate Fact-to-Fact Rules.** For each predicate symbol and its specified arity, we generate a set of candidate Horn clauses with that predicate as the head (consequent). For the rule body (antecedent), we consider all conjunctions of literals involving other predicates (i.e., we do not allow recursive rules) up to a fixed maximum length. Each candidate rule is scored on the mentions in the training documents for *support* (number of training examples that mention all facts in the body) and *confidence* (the conditional probability that the head is mentioned given that the body is satisfied). We discard all rules that do not achieve minimum support $\sigma$ and then keep the top $\tau$ most confident rules. The values of $\sigma$ and $\tau$ are determined via cross-validation within the training set. The selected rules are then entered into the knowledge base. From each fact-to-fact rule, we derive mention-to-mention rules as described above. For each predicate, we also generate fact-to-mention and mention-to-fact rules.

**Learning the Weights.** The goal of weight learning is to maximize the likelihood of the observed mentions (in the training set) by adjusting the weights of the rules. Because our training data only consists of mentions and no facts, the facts are latent (hidden variables), and we must apply the EM algorithm to learn the weights.

We employ the Markov Logic system Alchemy [8] for learning and inference. To implement EM, we applied the MC-SAT algorithm in the E-step and maximum pseudo-log likelihood ("generative training") for the M step. EM is iterated to convergence, which only requires a few iterations. Table 2 summarizes the pseudo-code of the algorithm. MAP inference for prediction is achieved using Alchemy's extension of MaxWalkSat.

**Treating Missing Mentions as Missing At Random:** An alternative to the Gricean mention model described above is to assume that the writer chooses which facts to mention (or omit) at random

Table 2: Learn Gricean Mention Model

**Input**: $\mathcal{D}_I$ =Incomplete training examples
$\tau$ = number of rules per head
$\sigma$ = minimum support per rule
**Output**: $\mathcal{M}$ = Explicit mention model

```
 1: LEARN GRICEAN MENTION MODEL:
 2: exhaustively learn rules for each head
 3: discard rules with less than σ support
 4: select the τ most confident rules R for each head
 5: R' := R
 6: for each rule (factP => factQ) ∈ R do
 7:     add mentionP ⇒ ¬mentionQ to R'
 8: end for
 9: for every factP ∈ R do
10:     add factP ⇒ mentionP to R'
11:     add mentionP ⇒ factP to R'
12: end for
13: repeat
14:     E-Step: apply inference to predict weighted facts F
15:     define complete weighted data D_C := D_I ∪ F
16:     M-Step: learn weights for rules in R' using data D_C
17: until convergence
18: return the set of weighted rules R'
```

Table 3: Synthetic Data Properties

|  |  | | | | $q$ | | |
|---|---|---|---|---|---|---|---|
|  |  | 0.17 | 0.33 | 0.50 | 0.67 | 0.83 | 0.97 |
| Mentioned literals | (%) | 91.38 | 80.74 | 68.72 | 63.51 | 51.70 | 42.13 |
| Complete records | (%) | 61.70 | 30.64 | 8.51 | 5.53 | 0.43 | 0.00 |

according to some unknown probability distribution that does not depend on the values of the missing variables—a setting known as Missing-At-Random (MAR). When data are MAR, it is possible to obtain unbiased estimates of the true distribution via imputation using EM [16]. We implemented this approach as follows. We apply the same method of learning rules (requiring minimum support $\sigma$ and then taking the $\tau$ most confident rules). Each learned rule has the general form MENTION_A $\Rightarrow$ MENTION_B. The collection of rules is treated as a model of the joint distribution over the mentions. Generative weight learning combined with Alchemy's builtin EM implementation is then applied to learn the weights on these rules.

## 3 Experimental Evaluation

We evaluated our mention model approach using data generated from a known mention model to understand its behavior. Then we compared its performance to the MAR approach on actual extractions from news stories about NFL football games, citizenship, and Somali ship hijackings.

**Synthetic Mention Experiment.** The goal of this experiment was to evaluate the ability of our method to learn accurate rules from data that match the assumptions of the algorithm. We also sought to understand how performance varies as a function of the amount of information omitted from the text.

The data were generated using a database of NFL games (from 1998 and 2000-2005) downloaded from www.databasefootball.com. These games were then encoded using the predicates TEAMINGAME($Game, Team$), GAMEWINNER($Game, Team$), GAMELOSER($Game, Team$), HOMETEAM($Game, Team$), AWAYTEAM($Game, Team$), and TEAMGAMESCORE($Game, Team, Score$) and treated as the ground truth. Note that these predicates can be divided into two correlated sets: $WL = \{$GAMEWINNER, GAMELOSER, TEAMGAMESCORE$\}$ and $HA = \{$HOMETEAM, AWAYTEAM$\}$.

From this ground truth, we generate a set of mentions for each game as follows. One literal is chosen uniformly at random from each of $WL$ and $HA$ and mentioned. Then each of the remaining literals is mentioned with probability $1-q$, where $q$ is a parameter that we varied in the experiments. Table 3 shows the average percentage of literals mentioned in each generated "news story" and the percentage of generated "news stories" that mentioned all literals.

For each $q$, we generated 5 different datasets, each containing 235 games. For each value of $q$, we ran the algorithm five times. In each iteration, one dataset was used for training, another for validation, and the remaining 3 for testing. The training and validation datasets shared the same value of $q$. The resulting learned rules were evaluated on the test sets for all of the different values of $q$. The validation set is employed to determine the thresholds $\tau$ and $\sigma$ during rule learning and to decide when to terminate EM. The chosen values were $\tau = 10$, $\sigma = 0.5$ (50% of the total training instances), and between 3 and 8 EM iterations.

Table 4: Gricean Mention Model Performance on Synthetic Data. Each cell indicates % of complete records inferred.

| **Training** $q$ | **Test** $q$ | | | | | |
|---|---|---|---|---|---|---|
|  | 0.17 (%) | 0.33 (%) | 0.50 (%) | 0.67 (%) | 0.83 (%) | 0.97 (%) |
| 0.17 | 100 | 100 | 100 | 100 | 100 | 100 |
| 0.33 | 100 | 99 | 97 | 96 | 90 | 85 |
| 0.50 | 100 | 99 | 98 | 97 | 93 | 87 |
| 0.67 | 100 | 98 | 92 | 92 | 81 | 66 |
| 0.83 | 99 | 98 | 72 | 71 | 61 | 54 |
| 0.97 | 91 | 81 | 72 | 68 | 56 | 41 |

Table 4 reports the proportion of complete game records (i.e., all four literals) that were correctly inferred, averaged over the five runs. Note that any facts mentioned in the generated articles are

automatically correctly inferred, so if no inference was performed at all, the results would match the second row of Table 3. Notice that when trained on data with low missingness (e.g. $q = 0.17$), the algorithm was able to learn rules that predict well for articles with much higher levels of missing values. This is because $q = 0.17$ means that only 8.62% of the literals are missing in the training dataset, which results in 61.70% complete records. These are sufficient to allow learning highly-accurate rules. However, as the proportion of missing literals in the training data increases, the algorithm starts learning incorrect rules, so performance drops. In particular, when $q = 0.97$, the training documents contain *no* complete records (Table 3). Nonetheless, the learned rules are still able to completely and correctly reconstruct 41% of the games!

The rules learned under such high levels of missingness are not totally correct. Here is an example of one learned rule (for $q = 0.97$):

$$\text{FACT\_HOMETEAM}(g, t1) \wedge \text{FACT\_TEAMINGAME}(g, t1) \Rightarrow \text{FACT\_GAMEWINNER}(g, t1).$$

This rule says that the home team always wins. When appropriately weighted in Markov Logic, this is a reasonable rule even though it is not perfectly correct (nor was it a rule that we applied during the synthetic data generation process).

In addition to measuring the fraction of entire games correctly inferred, we can obtain a more fine-grained assessment by measuring the fraction of individual literals correctly inferred. Table 5 shows this for the $q = 0.97$ training scenario. We can see that even when the test articles

Table 5: Percentage of Literals Correctly Predicted

| Training $q$ | Test $q$ | | | | | |
|---|---|---|---|---|---|---|
| | 0.17 (%) | 0.33 (%) | 0.50 (%) | 0.67 (%) | 0.83 (%) | 0.97 (%) |
| 0.97 | 98 | 95 | 93 | 92 | 89 | 85 |

have $q = 0.97$ (which means only 42.13% of literals are mentioned), the learned rules are able to correctly infer 85% of the literals. By comparison, if the literals had been predicted independently at random, only 6.25% would be correctly predicted.

**Experiments with Real Data:** We performed experiments on three datasets extracted from news stories: NFL games, citizenship, and Somali ship hijackings.

**NFL Games.** A state-of-the-art information extraction system from BBN Technologies [6, 14] was applied to a corpus of 1000 documents taken from the Gigaword corpus V4 [13] to extract the same five propositions employed in the synthetic data experiments. The BBN coreference system attempted to detect and combine multiple mentions of the same game within a single article. The resulting data set contained 5,850 games. However, the data still contained many coreference errors, which produced games apparently involving more than two teams or where one team achieved multiple scores.

Table 6: Statistics on mentions for extracted NFL games (after repairing violations of integrity constraints). Under "Home/Away", "men none" gives the percentage of articles in which neither the Home nor the Away team was mentioned, "men one", the percentage in which exactly one of Home or Away was mentioned, and "men both", the percentage where both were mentioned.

| | Home/Away | | | Winner/Loser | | |
|---|---|---|---|---|---|---|
| | men none (%) | men one (%) | men both (%) | men none (%) | men one (%) | men both (%) |
| NFL Train | 17.9 | 58.9 | 23.2 | 17.9 | 57.1 | 25.0 |
| NFL Test | 83.6 | 19.6 | 0.0 | 1.8 | 98.2 | 0.0 |

To address these problems, we took each extracted game and applied a set of integrity constraints. The integrity constraints were learned automatically from 5 complete game records. Examples of the learned constraints include "Every game has exactly two teams" and "Every game has exactly one winner." Each extracted game was then converted into multiple games by deleting literals in all possible ways until all of the integrity constraints were satisfied. The team names were replaced (arbitrarily) with constants $A$ and $B$. The games were then processed to remove duplicates. The result was a set of 56 distinct extracted games, which we call **NFL Train**. To develop a test set, **NFL Test**, we manually extracted 55 games from news stories about the 2010 NFL season (which has no overlap with Gigaword V4). Table 6 summarizes these game records.

Here is an excerpt from one of the stories that was analyzed during learning: *"William Floyd rushed for three touchdowns and Steve Young scored two more, moving the San Francisco 49ers one victory*

*from the Super Bowl with a 44-15 American football rout of Chicago."* The initial set of literals extracted by the BBN system was the following:

MENTION_TEAMINGAME($NFLGame9209, SanFrancisco49ers$) ∧
MENTION_TEAMINGAME($NFLGame9209, ChicagoBears$) ∧
MENTION_GAMEWINNER($NFLGame9209, SanFrancisco49ers$) ∧
MENTION_GAMEWINNER($NFLGame9209, ChicagoBears$) ∧
MENTION_GAMELOSER($NFLGame9209, ChicagoBears$).

After processing with the learned integrity constraints, the extracted interpretation was the following:

MENTION_TEAMINGAME($NFLGame9209, SanFrancisco49ers$) ∧
MENTION_TEAMINGAME($NFLGame9209, ChicagoBears$) ∧
MENTION_GAMEWINNER($NFLGame9209, SanFrancisco49ers$) ∧
MENTION_GAMELOSER($NFLGame9209, ChicagoBears$).

It is interesting to ask whether these data are consistent with the explicit mention model versus the missing-at-random model. Let us suppose that under MAR, the probability that a fact will be mentioned is $p$. Then the probability that both literals in a rule (e.g., home/away or winner/loser) will be mentioned is $p^2$, the probability that both will be missing is $(1-p)^2$, and the probability that exactly one will be mentioned is $2p(1-p)$. We can fit the best value for $p$ to the observed missingness rates to minimize the KL divergence between the predicted

Table 7: Observed percentage of cases where exactly one literal is mentioned and the percentage predicted if the literals were missing at random

| | Home/Away | | Winner/Loser | |
|---|---|---|---|---|
| | obs. men one (%) | pred. men one (%) | obs. men one (%) | pred. men one (%) |
| NFL Train | 58.9 | 49.9 | 57.1 | 49.8 |
| NFL Test | 19.6 | 34.5 | 98.2 | 47.9 |

and observed distributions. If the explicit mention model is correct, then the MAR fit will be a poor estimate of the fraction of cases where exactly one literal is missing. Table 7 shows the results. On NFL Train, it is clear that the MAR model seriously underestimates the probability that exactly one literal will be mentioned. The NFL Test data is inconsistent with the MAR assumption, because there are no cases where both predicates are mentioned. If we estimate $p$ based only on the cases where both are missing or one is missing, the MAR model seriously underestimates the one-missing probability. Hence, we can see that train and test, though drawn from different corpora and extracted by different methods, both are inconsistent with the MAR assumption.

We applied both our explicit mention model and the MAR model to the NFL dataset. The cross-validated parameter values for the explicit mention model were $\epsilon = 0.5$ and $\tau = 50$, and the number of EM iterations varied between 2 and 3. We measured performance relative to the performance that could be attained by a system that uses the correct rules. The results are summarized in Table 8. Our method achieves perfect performance, whereas the MAR method only reconstructs half of the reconstructable games. This reflects the extreme difficulty of the test set, where none of the articles mentions all literals involved in any rule.

Table 8: NFL test set performance.

| Gricean Model (%) | MAR Model (%) |
|---|---|
| 100.0 | 50.0 |

Here are a few examples of the rules that are learned:

$0.00436$ : FACT_TEAMINGAME($g, t_1$) ∧ FACT_GAMELOSER($g, t_2$) ∧ ($t_1 \neq t_2$) ⇒
FACT_GAMEWINNER($g, t_1$)
$0.17445$ : MENTION_TEAMINGAME($g, t_1$) ∧ MENTION_GAMELOSER($g, t_2$) ∧ ($t_1 \neq t_2$) ⇒
¬MENTION_GAMEWINNER($g, t_1$)

The first rule is a weak form of the "fact" rule that if one team is the loser, the other is the winner. The second rule is the corresponding "mention" rule that if the loser is mentioned then the winner is not. The small weights on these rules are difficult to interpret in isolation, because in Markov logic, all of the weights are coupled and there are other learned rules that involve the same literals.

**Birthplace and Citizenship**. We repeated this same experiment on a different set of 182 articles selected from the ACE08 Evaluation corpus [12] and extracted by the same methods. In these

articles, the citizenship of a person is mentioned 583 times and birthplace only 25 times. Both are mentioned in the same article only 6 times (and of these, birthplace and citizenship are the same in only 4). Clearly, this is another case where the MAR assumption does not hold. Integrity constraints were applied to force each person to have at most one birthplace and one country of citizenship, and then both methods were applied. The cross-validated parameter values for the explicit mention model were $\epsilon = 0.5$ and $\tau = 50$ and the number of EM iterations varied between 2 and 3. Table 9 shows the two cases of interest and the probability assigned to the missing fact by the two methods. The inverse Gricean approach gives much better results.

**Somali Ship Hijacking**. We collected a set of 41 news stories concerning ship hijackings based on ship names taken from the web site `coordination-maree-noire.eu`. From these documents, we manually extracted all mentions of the ownership country and flag country of the hijacked ships. Twenty-five stories mentioned only one fact (ownership or flag), while 16 mentioned both.

Table 9: Birthplace and Citizenship: Predicted probability assigned to the correct interpretation by the Gricean mention model and the MAR model.

| Configuration | Gricean Model Pred. prob. | MAR Pred. prob. |
|---|---|---|
| Citizenship missing | 1.000 | 0.969 |
| Birthplace missing | 1.000 | 0.565 |

Of the 16, 14 reported the flag country as different from the ownership country. The Gricean maxims predict that if the two countries are the same, then only one of them will be mentioned. The results (Table 10) show that the Gricean model is again much more accurate than the MAR model.

## 4 Conclusion

This paper has shown how to formalize the Gricean conversational maxims, compile them into Markov Logic, and invert them via probabilistic reasoning to learn Horn clause rules from facts extracted from documents. Experiments on synthetic mentions showed that our method is able to correctly reconstruct complete records even when neither the training data nor the test data contain complete records. Our three studies provide evidence that news articles obey the maxims

Table 10: Flag and Ownership: Predicted probability assigned to the missing fact by the Gricean mention model and the MAR model. Cross-validated parameter values $\epsilon = 0.5$ and $\tau = 50$; 2-3 EM iterations.

| Configuration | Gricean Model Pred. prob. | MAR Pred. prob. |
|---|---|---|
| Ownership missing | 1.000 | 0.459 |
| Flag missing | 1.000 | 0.519 |

across three domains. In all three domains, our method achieves excellent performance that far exceeds the performance of standard EM imputation. This shows conclusively that rule learning benefits from employing an explicit model of the process that generates the data. Indeed, it allows rules to be learned correctly from only a handful of complete training examples.

An interesting direction for future work is to learn forms of knowledge more complex than Horn clauses. For example, the state of a hijacked ship can change over time from states such as "attacked" and "captured" to states such as "ransom demanded" and "released". The Gricean mention model predicts that if a news story mentions that a ship was released, then it does not need to mention that the ship was "attacked" or "captured". Handling such cases will require extending the methods in this paper to reason about time and what the author and reader know at each point in time. It will also require better methods for joint inference, because there are more than 10 predicates in this domain, and our current EM implementation scales exponentially in the number of interrelated predicates.

**Acknowledgments**

This material is based upon work supported by the Defense Advanced Research Projects Agency (DARPA) under Contract No. FA8750-09-C-0179 and by Army Research Office (ARO). Any opinions, findings and conclusions or recommendations expressed in this material are those of the author(s) and do not necessarily reflect the views of the DARPA, the Air Force Research Laboratory (AFRL), ARO, or the US government.

# References

[1] A. Carlson, J. Betteridge, B. Kisiel, B. Settles, E.R. Hruschka Jr., and T.M. Mitchell. Toward an architecture for never-ending language learning. In *Proceedings of the Conference on Artificial Intelligence (AAAI)*, pages 1306–1313. AAAI Press, 2010.

[2] A. Carlson, J. Betteridge, R. C. Wang, E. R. Hruschka, Jr., and T. M. Mitchell. Coupled semi-supervised learning for information extraction. In *Proceedings of the Third ACM International Conference on Web Search and Data Mining*, WSDM '10, pages 101–110, New York, NY, USA, 2010. ACM.

[3] W. W. Cohen. WHIRL: A word-based information representation language. *Artificial Intelligence*, 118(1-2):163–196, 2000.

[4] J. R. Doppa, M. S. Sorower, M. Nasresfahani, J. Irvine, W. Orr, T. G. Dietterich, X. Fern, and P. Tadepalli. Learning rules from incomplete examples via implicit mention models. In *Proceedings of the 2011 Asian Conference on Machine Learning*, 2011.

[5] O. Etzioni, M. Banko, S. Soderland, and D. S. Weld. Open information extraction from the web. *Commun. ACM*, 51(12):68–74, 2008.

[6] M. Freedman, E. Loper, E. Boschee, and R. Weischedel. Empirical Studies in Learning to Read. In *Proceedings of Workshop on Formalisms and Methodology for Learning by Reading (NAACL-2010)*, pages 61–69, 2010.

[7] H. P. Grice. Logic and conversation. In *Syntax and semantics: Speech acts*, volume 3, pages 43–58. Academic Press, New York, 1975.

[8] S. Kok, M. Sumner, M. Richardson, P. Singla, H. Poon, D. Lowd, and P. Domingos. The Alchemy system for statistical relational AI. *Technical report, Department of Computer Science and Engineering, University of Washington, Seattle, WA*, 2007.

[9] L. Michael. Reading between the lines. In *IJCAI*, pages 1525–1530, 2009.

[10] L. Michael and L. G. Valiant. A first experimental demonstration of massive knowledge infusion. In *KR*, pages 378–389, 2008.

[11] U. Y. Nahm and R. J. Mooney. A mutually beneficial integration of data mining and information extraction. In *Proceedings of the Seventeenth National Conference on Artificial Intelligence and the Twelfth Conference on Innovative Applications of Artificial Intelligence*, pages 627–632. AAAI Press, 2000.

[12] NIST. Automatic Content Extraction 2008 Evaluation Plan.

[13] R. Parker, D. Graff, J. Kong, K. Chen, and K. Maeda. *English Gigaword Fourth Edition*. Linguistic Data Consortium, Philadelphia, 2009.

[14] L. Ramshaw, E. Boschee, M. Freedman, J. MacBride, R. Weischedel, and A.Zamanian. Serif language processing effective trainable language understanding. In Joseph Olive, Caitlin Christianson, and John McCary, editors, *Handbook of Natural Language Processing and Machine Translation: DARPA Global Autonomous Language Exploitation*. Springer, 2011.

[15] M. Richardson and P. Domingos. Markov logic networks. *Machine learning*, 62:107–136, February 2006.

[16] J. L. Schafer and M. K. Olsen. Multiple imputation for multivariate missing-data problems: a data analyst's perspective. *Multivariate Behavioral Research*, 33:545–571, 1998.

[17] S. Schoenmackers, O. Etzioni, D. S. Weld, and J. Davis. Learning first-order Horn clauses from web text. In *Proceedings of the 2010 Conference on Empirical Methods in Natural Language Processing*, EMNLP '10, pages 1088–1098, Stroudsburg, PA, USA, 2010. Association for Computational Linguistics.

